# Value Pursuit Iteration

**Amir-massoud Farahmand**[*]          **Doina Precup** [†]
School of Computer Science, McGill University, Montreal, Canada

## Abstract

Value Pursuit Iteration (VPI) is an approximate value iteration algorithm that finds a close to optimal policy for reinforcement learning problems with large state spaces. VPI has two main features: First, it is a nonparametric algorithm that finds a good sparse approximation of the optimal value function given a dictionary of features. The algorithm is almost insensitive to the number of irrelevant features. Second, after each iteration of VPI, the algorithm adds a set of functions based on the currently learned value function to the dictionary. This increases the representation power of the dictionary in a way that is directly relevant to the goal of having a good approximation of the optimal value function. We theoretically study VPI and provide a finite-sample error upper bound for it.

## 1 Introduction

One often has to use function approximation to represent the near optimal value function of the reinforcement learning (RL) and planning problems with large state spaces. Even though the conventional approach of using a parametric model for the value function has had successes in many applications, it has one main weakness: Its success critically depends on whether the chosen function approximation method is suitable for the particular task in hand. Manually designing a suitable function approximator, however, is difficult unless one has considerable domain knowledge about the problem. To address this issue, the problem-dependent choice of function approximator and nonparametric approaches to RL/Planning problems have gained considerable attention in the RL community, e.g., feature generation methods of Mahadevan and Maggioni [1], Parr et al. [2], and nonparametric regularization-based approaches of Farahmand et al. [3, 4], Taylor and Parr [5] .

One class of approaches that addresses the aforementioned problem is based on the idea of finding a sparse representation of the value function in a large dictionary of features (or *atoms*). In this approach, the designer does not necessarily know a priori whether or not a feature is relevant to the representation of the value function. The feature, therefore, is simply added to the dictionary with the hope that the algorithm itself figures out the necessary subset of features. The usual approach to tackle irrelevant features is to use sparsity-inducing regularizers such as the $l_1$-norm of the weights in the case of linear function approximators, e.g., Kolter and Ng [6], Johns et al. [7], Ghavamzadeh et al. [8]. Another approach is based on greedily adding atoms to the representation of the target function. Examples of these approaches in the supervised learning setting are Matching Pursuit and Orthogonal Matching Pursuit (OMP) [9, 10]. These greedy algorithms have successfully been used in the signal processing and statistics/supervised machine learning communities for years, but their application in the RL/Planning problems has just recently attracted some attention. Johns [11] empirically investigated some greedy algorithms, including OMP, for the task of feature selection using dictionary of proto-value functions [1]. A recent paper by Painter-Wakefield and Parr [12] considers two algorithms (OMP-TD and OMP-BRM; OMP-TD is the same as one of the algorithms by [11]) in the context of policy evaluation and provides some conditions under which OMP-BRM can find the minimally sparse solution.

---

[*] `Academic.SoloGen.net`.

[†] This research was funded in part by NSERC and ONR.

To address the problem of value function representation in RL when not much a priori knowledge is available, we introduce the **Value Pursuit Iteration (VPI)** algorithm. VPI, which is an Approximate Value Iteration (AVI) algorithm (e.g., [13]), has two main features. The first is that it is a nonparametric algorithm that finds a good sparse approximation of the optimal value function given a set of features (dictionary), by using a modified version of OMP. The second is that after each iteration, the VPI algorithm adds a set of functions based on the currently learned value function to the dictionary. This potentially increases the representation power of the dictionary in a way that is directly relevant to the goal of approximating the optimal value function.

At the core of VPI is the OMP algorithm equipped with a model selection procedure. Using OMP allows VPI to find a sparse representation of the value function in large dictionaries, even countably infinite ones[1]. This property is very desirable for RL/Planning problems for which one usually does not know the right representation of the value function, and so one wishes to add as many features as possible and to let the algorithm automatically detect the best representation. A model selection procedure ensures that OMP is adaptive to the actual difficulty of the problem.

The second main feature of VPI is that it increases the size of the dictionary by adding some basis functions computed from previously learned value functions. To give an intuitive understanding of how this might help, consider the dictionary $\mathcal{B} = \{g_1, g_2, \dots\}$, in which each atom $g_i$ is a real-valued function defined on the state-action space. The goal is to learn the optimal value function by a representation in the form of $Q = \sum_{i \geq 1} w_i g_i$.[2] Suppose that we are lucky and the optimal value function $Q^*$ belongs to the dictionary $\mathcal{B}$, e.g., $g_1 = Q^*$. This is indeed an ideal atom to have in the dictionary since one may have a sparse representation of the optimal value function in the form of $Q^* = \sum_{i \geq 1} w_i g_i$ with $w_1 = 1$ and $w_i = 0$ for $i \geq 2$. Algorithms such as OMP can find this sparse representation quite effectively (details will be specified later). Of course, we are not usually lucky enough to have the optimal value function in our dictionary, but we may still use approximation of the optimal value function. In the exact Value Iteration, $Q_k \to Q^*$ exponentially fast. This ensures that $Q_k$ and $Q_{k+1} = T^*Q_k$ are close enough, so one may use $Q_k$ to explain a large part of $Q_{k+1}$ and use the other atoms of the dictionary to "explain" the residual. In an AVI procedure, however, the estimated value function sequence $(Q_k)_{k \geq 1}$ does not necessarily converge to $Q^*$, but one may hope that it gets close to a region around the optimum. In that case, we may very well use the dictionary of $\{Q_1, \dots, Q_k\}$ as the set of candidate atoms to be used in the representation of $Q_{k+1}$. We show that adding these learned atoms does not hurt and may actually help.

To summarize, the algorithmic contribution of this paper is to introduce the VPI algorithm that finds a sparse representation of the optimal value function in a huge function space and increases the representation capacity of the dictionary problem-dependently. The theoretical contribution of this work is to provide a finite-sample analysis of VPI and to show that the method is sound.

## 2    Definitions

We follow the standard notation and definitions of Markov Decision Processes (MDP) and Reinforcement Learning (RL) (cf. [14]). We also need some definitions regarding the function spaces and norms, which are defined later in this section.

For a space $\Omega$ with $\sigma$-algebra $\sigma_\Omega$, $\mathcal{M}(\Omega)$ denotes the set of all probability measures over $\sigma_\Omega$. $B(\Omega)$ denotes the space of bounded measurable functions w.r.t. (with respect to) $\sigma_\Omega$ and $B(\Omega, L)$ denotes the subset of $B(\Omega)$ with bound $0 < L < \infty$.

A *finite-action discounted MDP* is a 5-tuple $(\mathcal{X}, \mathcal{A}, P, \mathcal{R}, \gamma)$, where $\mathcal{X}$ is a measurable state space, $\mathcal{A}$ is a finite set of actions, $P : \mathcal{X} \times \mathcal{A} \to \mathcal{M}(\mathcal{X})$ is the transition probability kernel, $\mathcal{R} : \mathcal{X} \times \mathcal{A} \to \mathcal{M}(\mathbb{R})$ is the reward kernel, and $\gamma \in [0, 1)$ is a discount factor. Let $r(x, a) = \mathbb{E}\left[\mathcal{R}(\cdot|x, a)\right]$, and assume that $r$ is uniformly bounded by $R_{\max}$. A measurable mapping $\pi : \mathcal{X} \to \mathcal{A}$ is called a deterministic Markov stationary policy, or just a *policy* for short. A policy $\pi$ induces the $m$-step transition probability kernels $(P^\pi)^m : \mathcal{X} \to \mathcal{M}(\mathcal{X})$ and $(P^\pi)^m : \mathcal{X} \times \mathcal{A} \to \mathcal{M}(\mathcal{X} \times \mathcal{A})$ for $m \geq 1$.

We use $V^\pi$ and $Q^\pi$ to denote the value and action-value function of a policy $\pi$. We also use $V^*$ and $Q^*$ for the optimal value and optimal action-value functions, with the corresponding optimal

policy $\pi^*$. A policy $\pi$ is *greedy* w.r.t. an action-value function $Q$, denoted $\pi = \hat{\pi}(\cdot; Q)$, if $\pi(x) = \operatorname{argmax}_{a \in \mathcal{A}} Q(x, a)$ holds for all $x \in \mathcal{X}$ (if there exist multiple maximizers, one of them is chosen in an arbitrary deterministic manner). Define $Q_{\max} = R_{\max}/(1 - \gamma)$. The Bellman optimality operator is denoted by $T^*$. We use $(PV)(x)$ to denote the expected value of $V$ after the transition according to a probability transition kernel $P$. Also for a probability measure $\rho \in \mathcal{M}(\mathcal{X})$, the symbol $(\rho P)$ represents the distribution over states when the initial state distribution is $\rho$ and we follow $P$ for a single step. A typical choice of $P$ is $(P^\pi)^m$ for $m \geq 1$ (similarly for $\rho \in \mathcal{M}(\mathcal{X} \times \mathcal{A})$ and action-value functions).

## 2.1 Norms and Dictionaries

For a probability measure $\rho \in \mathcal{M}(\mathcal{X})$, and a measurable function $V \in B(\mathcal{X})$, we define the $L_p(\rho)$-norm ($1 \leq p < \infty$) of $V$ as $\|V\|_{p,\rho} \triangleq \left[ \int_{\mathcal{X}} |V(x)|^p \, \mathrm{d}\rho(x) \right]^{1/p}$. The $L_\infty(\mathcal{X})$-norm is defined as $\|V\|_\infty \triangleq \sup_{x \in \mathcal{X}} |V(x)|$. Similarly for $\nu \in \mathcal{M}(\mathcal{X} \times \mathcal{A})$ and $Q \in B(\mathcal{X} \times \mathcal{A})$, we define $\|\cdot\|_{p,\nu}$ as $\|Q\|_{p,\nu}^p \triangleq \int_{\mathcal{X} \times \mathcal{A}} |Q(x,a)|^p d\nu(x,a)$ and $\|Q\|_\infty \triangleq \sup_{(x,a) \in \mathcal{X} \times \mathcal{A}} |Q(x,a)|$.

Let $z_{1:n}$ denote the $\mathcal{Z}$-valued sequence $(z_1, \ldots, z_n)$. For $\mathcal{D}_n = z_{1:n}$, define the empirical norm of function $f : \mathcal{Z} \to \mathbb{R}$ as $\|f\|_{p,z_{1:n}}^p = \|f\|_{p,\mathcal{D}_n}^p \triangleq \frac{1}{n} \sum_{i=1}^n |f(z_i)|^p$. Based on this definition, one may define $\|V\|_{\mathcal{D}_n}$ (with $\mathcal{Z} = \mathcal{X}$) and $\|Q\|_{\mathcal{D}_n}$ (with $\mathcal{Z} = \mathcal{X} \times \mathcal{A}$). Note that if $\mathcal{D}_n = Z_{1:n}$ is random with $Z_i \sim \nu$, the empirical norm is random as well. For any fixed function $f$, we have $\mathbb{E}\left[ \|f\|_{p,\mathcal{D}_n} \right] = \|f\|_{p,\nu}$. The symbols $\|\cdot\|_\nu$ and $\|\cdot\|_{\mathcal{D}_n}$ refer to an $L_2$-norm. When we do not want to emphasize the underlying measure, we use $\|\cdot\|$ to denote an $L_2$-norm.

Consider a Hilbert space $\mathcal{H}$ endowed with an inner product norm $\|\cdot\|$. We call a family of functions $\mathcal{B} = \{g_1, g_2, \ldots, \}$ with atoms $g_i \in \mathcal{H}$ a *dictionary*. The class $\mathcal{L}_1(\mathcal{B}) = \mathcal{L}_1(\mathcal{B}; \|\cdot\|)$ consists of those functions $f \in \mathcal{H}$ that admits an expansion $f = \sum_{g \in \mathcal{B}} c_g g$ with $(c_g)$ being an absolutely summable sequence (these definitions are quoted from Barron et al. [15]). The norm of a function $f$ in this space is defined as $\|f\|_{\mathcal{L}_1(\mathcal{B}; \|\cdot\|)} \triangleq \inf\{\sum_{g \in \mathcal{B}} |c_g| : f = \sum_{g \in \mathcal{B}} c_g g\}$. To avoid clutter, when the norm is the empirical norm $\|\cdot\|_{\mathcal{D}_n}$, we may use $\mathcal{L}_1(\mathcal{B}; \mathcal{D}_n)$ instead of $\mathcal{L}_1(\mathcal{B}; \|\cdot\|_{\mathcal{D}_n})$, and when the norm is $\|\cdot\|_\nu$, we may use $\mathcal{L}_1(\mathcal{B}; \nu)$. We denote a ball with radius $r > 0$ w.r.t. the norm of $\mathcal{L}_1(\mathcal{B}; \nu)$ by $B_r(\mathcal{L}_1(\mathcal{B}; \nu))$.

For a dictionary $\mathcal{B}$, we introduce a fixed exhaustion $\mathcal{B}_1 \subset \mathcal{B}_2 \subset \ldots \subset \mathcal{B}$, with the number of atoms $|\mathcal{B}_m|$ being $m$. If we index our dictionaries as $\mathcal{B}_k$, the symbol $\mathcal{B}_{k,m}$ refers to the $m$-th element of the exhaustion of $\mathcal{B}_k$. For a real number $\alpha > 0$, the space $\mathcal{L}_{1,\alpha}(\mathcal{B}; \|\cdot\|)$ is defined as the set of all functions $f$ such that for all $m = 1, 2, \ldots$, there exists a function $h$ depending on $m$ such that $\|h\|_{\mathcal{L}_1(\mathcal{B}_m; \|\cdot\|)} \leq C$ and $\|f - h\| \leq C m^{-\alpha}$. The smallest constant $C$ such that these inequalities hold defines a norm for $\mathcal{L}_{1,\alpha}(\mathcal{B}; \|\cdot\|)$. Finally, we define the truncation operator $\beta_L : B(\mathcal{X}) \to B(\mathcal{X})$ for some real number $L > 0$ as follows. For any function $f \in B(\mathcal{X})$, the truncated function of $f$ at the threshold level $L$ is the function $\beta_L f : B(\mathcal{X}) \to \mathbb{R}$ such that for any $x \in \mathcal{X}$, $\beta_L f(x)$ is equal to $f(x)$ if $-L \leq f(x) \leq L$, is equal to $L$ if $f(x) > L$, and is equal to $-L$ if $f(x) < -L$. We overload $\beta_L$ to be an operator from $B(\mathcal{X} \times \mathcal{A})$ to $B(\mathcal{X} \times \mathcal{A})$ by applying it component-wise, i.e., $\beta_L Q(x, \cdot) \triangleq [\beta_L Q(x, a_1), \ldots, \beta_L Q(x, a_\mathcal{A})]^\top$.

## 3 VPI Algorithm

In this section, we first describe the behaviour of VPI in the ideal situation when the Bellman optimality operator $T^*$ can be applied exactly in order to provide the intuitive understanding of why VPI might work. Afterwards, we describe the algorithm that does not have access to the Bellman optimality operator and only uses a finite sample of transitions.

VPI belongs to the family of AVI algorithms, which start with an initial action-value function $Q_0$ and at each iteration $k = 0, 1, \ldots$, approximately apply the Bellman optimality operator $T^*$ to the most recent estimate $Q_k$ to get a new estimate $Q_{k+1} \approx T^* Q_k$. The size of the error between $Q_{k+1}$ and $T^* Q_k$ is a key factor in determining the performance of an AVI procedure.

Suppose that $T^*Q_k$ can be calculated, but it is not possible to represent it exactly. In this case, one may use an approximant $Q_{k+1}$ to represent $T^*Q_k$. In this paper we would like to represent $Q_{k+1}$ as a linear function of some atoms in a dictionary $\mathcal{B} = \{g_1, g_2, \dots\}$ ($g \in \mathcal{H}(\mathcal{X} \times \mathcal{A})$ and $\|g\| = 1$ for all $g \in \mathcal{B}$), that is $Q_{k+1} = \sum_{g \in \mathcal{B}} c_g g$. Our goal is to find a representation that is as sparse as possible, i.e., uses only a few atoms in $\mathcal{B}$. From statistical viewpoint, the smallest representation among all those that have the same function approximation error is desirable as it leads to smaller estimation error. The goal of finding the sparsest representation, however, is computationally intractable. Nevertheless, it is possible to find a "reasonable" suboptimal sparse approximation using algorithms such as OMP, which is the focus of this paper.

The OMP algorithm works as follows. Let $\tilde{Q}^{(0)} = 0$. For each $i = 1, 2, \dots$, define the residual $r^{(i-1)} = T^*Q_k - \tilde{Q}^{(i-1)}$. Define the new atom to be added to the representation as $g^{(i)} \in \operatorname{Argmax}_{g \in \mathcal{B}} \left| \left\langle r^{(i-1)}, g \right\rangle \right|$, i.e., choose an element of the dictionary that has the maximum correlation with the residual. Here $\langle \cdot, \cdot \rangle$ is the inner product for a Hilbert space $\mathcal{H}(\mathcal{X} \times \mathcal{A})$ to which $T^*Q_k$ and atoms of the dictionary belong. Let $\Pi^{(i)}$ be the orthogonal projection onto $\operatorname{span}(g^{(1)}, \dots, g^{(i)})$, i.e., $\Pi^{(i)}T^*Q_k \triangleq \operatorname{argmin}_{Q \in \operatorname{span}(g^{(1)}, \dots, g^{(i)})} \|Q - T^*Q_k\|$. We then have $\tilde{Q}^{(i)} = \Pi^{(i)}T^*Q_k$. OMP continues iteratively.

To quantify the approximation error at the $i$-th iteration, we use the $\mathcal{L}_1(\mathcal{B}; \|\cdot\|)$-norm of the target function of the OMP algorithm, which is $T^*Q_k$ in our case (with the norm being the one induced by the inner product used in the OMP procedure). Recall that this class consists of functions that admit an expansion in the form $\sum_{g \in \mathcal{B}} c_g g$ and $(c_g)$ being an absolutely summable sequence. If $T^*Q_k$ belongs to the class of $\mathcal{L}_1(\mathcal{B}; \|\cdot\|)$, it can be shown (e.g., Theorem 2.1 of Barron et al. [15]) that after $i$ iterations of OMP, the returned function $\tilde{Q}^{(i)}$ is such that $\|\tilde{Q}^{(i)} - T^*Q_k\| \leq \frac{\|T^*Q_k\|_{\mathcal{L}_1(\mathcal{B}; \|\cdot\|)}}{\sqrt{i+1}}$. The problem with this result is that it requires $T^*Q_k$ to belong to $\mathcal{L}_1(\mathcal{B}; \|\cdot\|)$. This depends on how expressive the dictionary $\mathcal{B}$ is. If it is not expressive enough, we still would like OMP to quickly converge to the best approximation of $T^*Q_k \notin \mathcal{L}_1(\mathcal{B}; \|\cdot\|)$ in $\mathcal{L}_1(\mathcal{B}; \|\cdot\|)$. Fortunately, such a result exists (Theorem 2.3 by Barron et al. [15], quoted in the supplementary material) and we use it in the proof of our main result.

We now turn to the more interesting case when we do not have access to $T^*Q_k$. Instead we are only given a set of transitions in the form of $\mathcal{D}_n^{(k)} = \{(X_i^{(k)}, A_i^{(k)}, R_i^{(k)}, X_i'^{(k)})\}_{i=1}^n$, where $(X_i^{(k)}, A_i^{(k)})$ are drawn from the sampling distribution $\nu \in \mathcal{M}(\mathcal{X} \times \mathcal{A})$, $X_i' \sim P(\cdot | X_i, A_i)$, and $R_i \sim \mathcal{R}(\cdot | X_i, A_i)$. Instead of using $T^*Q_k$, we use the empirical Bellman operator for the dataset $\mathcal{D}_n^{(k)}$. The operator is defined as follows.

**Definition 1** (Empirical Bellman Optimality Operator). *Let $\mathcal{D}_n = \{(X_1, A_1, R_1, X_1'), \dots, (X_n, A_n, R_n, X_n')\}$, defined similarly as above. Define the ordered multiset $S_n = \{(X_1, A_1), \dots, (X_n, A_n)\}$. The empirical Bellman optimality operator $\hat{T}^* : S_n \to \mathbb{R}^n$ is defined as $(\hat{T}^*Q)(X_i, A_i) \triangleq R_i + \gamma \max_{a'} Q(X_i', a')$ for $1 \leq i \leq n$.*

Since $\mathbb{E}\left[\hat{T}^*Q_k(X_i^{(k)}, A_i^{(k)}) \,\middle|\, Q_k, X_i^{(k)}, A_i^{(k)}\right] = T^*Q_k(X_i^{(k)}, A_i^{(k)})$, we can solve a regression problem and find an estimate for $Q_{k+1}$, which is close $T^*Q_k$. This regression problem is the core of the family of Fitted Q-Iteration (FQI) algorithms, e.g., [13, 4]. In this paper, the regression function at each iteration is estimated using a modified OMP procedure introduced by Barron et al. [15].

We are now ready to describe the VPI algorithm (Algorithm 1). It gets as input a predefined dictionary $\mathcal{B}_0$. This can be a dictionary of wavelets, proto-value functions, etc. The size of this dictionary can be countably infinite. It also receives an integer $m$, which specifies how many atoms of $\mathcal{B}_0$ should be used by the algorithm. This defines the effective dictionary $\mathcal{B}_{0,m}$. This value can be set to $m = \lceil n^a \rceil$ for some finite $a > 0$, so it can actually be quite large. VPI also receives $K$, the number of iterations, and $\nu$, the sampling distribution. For the simplicity of analysis, we assume that the sampling distribution is fixed, but in practice one may change this sampling distribution after each iteration (e.g., sample new data according to the latest policy). Finally, VPI gets a set of $m'$ link functions $\sigma_i : B(\mathcal{X} \times \mathcal{A}, Q_{\max}) \to B(\mathcal{X} \times \mathcal{A}, Q_{\max})$ for some $m'$ that is smaller than $m/K$. We describe the role of link functions shortly.

---

**Algorithm 1** Value Pursuit Iteration($\mathcal{B}_0, m, \{\sigma_i\}_{i=1}^{m'}, \nu, K$)

---

**Input:** Initial dictionary $\mathcal{B}_0$, Number of dictionary atoms used $m$, Link functions $\{\sigma_i\}_{i=1}^{m'}$, State-action distribution $\nu$, Number of iterations $K$.

**Return:** $Q_K$

$Q_0 \leftarrow 0$.

$\mathcal{B}'_0 \leftarrow \emptyset$.

**for** $k = 0, 1, \ldots, K-1$ **do**

$\quad$ Construct a dataset $\mathcal{D}_n^{(k)} = \left\{ (X_i^{(k)}, A_i^{(k)}, R_i^{(k)}, X_i'^{(k)}) \right\}_{i=1}^n$, $(X_i^{(k)}, A_i^{(k)}) \overset{\text{i.i.d.}}{\sim} \nu$

$\quad \hat{Q}_{k+1}^{(0)} \leftarrow 0$

$\quad$ // Orthogonal Matching Pursuit loop

$\quad$ Normalize elements of $\mathcal{B}_{0,m}$ and $\mathcal{B}'_k$ according to $\|\cdot\|_{\mathcal{D}_n^{(k)}}$ and call them $\hat{\mathcal{B}}_k$ and $\hat{\mathcal{B}}'_k$.

$\quad$ **for** $i = 1, 2, \ldots, c_1 n$ **do**

$\quad\quad r^{(i-1)} \leftarrow \hat{T}^* Q_k - \hat{Q}_{k+1}^{(i-1)}$

$\quad\quad g^{(i)} \leftarrow \text{Argmax}_{g \in \hat{\mathcal{B}}_k \bigcup \hat{\mathcal{B}}'_k} \left| \left\langle r^{(i-1)}, g \right\rangle_{\mathcal{D}_n^{(k)}} \right|$

$\quad\quad \hat{Q}_{k+1}^{(i)} \leftarrow \Pi^{(i)} \hat{T}^* Q_k \quad\quad\quad\quad\quad$ {$\Pi^{(i)}$: Projection onto $\text{span}(g^{(1)}, \ldots, g^{(i)})$}

$\quad$ **end for**

$\quad i^* \leftarrow \text{argmin}_{i \geq 1} \left\{ \left\| \beta_{Q_{\max}} \hat{Q}_{k+1}^{(i)} - \hat{T}^* Q_k \right\|_{\mathcal{D}_n^{(k)}}^2 + c_2(Q_{\max}) \frac{i \ln(n)}{n} \right\}$ {Complexity Regularization}

$\quad Q_{k+1} \leftarrow \hat{Q}_{k+1}^{(i^*)}$

$\quad \mathcal{B}'_{k+1} \leftarrow \mathcal{B}'_k \bigcup \{\sigma_i(\beta_{Q_{\max}} Q_{k+1}; \mathcal{B}_k \bigcup \mathcal{B}'_k)\}_{i=1}^{m'}$ {Extending the dictionary}

**end for**

---

At the $k$-th iteration of the algorithm, we perform OMP for $c_1 n$ iterations ($c_1 > 0$), similar to what is described above with the difference that instead of using $T^* Q_k$ as the target, we use $\hat{T}^* Q_k$ over empirical samples.[3] This means that we use the empirical inner product $\langle Q_1, Q_2 \rangle_{\mathcal{D}_n^{(k)}} \triangleq \frac{1}{n} \sum_{i=1}^n |Q_1(X_i, A_i) \cdot Q_2(X_i, A_i)|$ for $(X_i, A_i) \in \mathcal{D}_n^{(k)}$ and the empirical orthogonal projection.[4] The result would be a sequence $(\hat{Q}_{k+1}^{(i)})_{i \geq 0}$. Next, we perform a model selection procedure to choose the best candidate. This can be done in different ways such as using a separate dataset as a validation set. Here we use a complexity regularization technique that penalizes more complex estimates (those that have more atoms in their representation). Note that we use the truncated estimate $\beta_{Q_{\max}} \hat{Q}_{k+1}^{(i)}$ in the model selection procedure. This is required for the theoretical guarantees. The outcome of this model selection procedure will determine $Q_{k+1}$.

Finally we use link functions $\{\sigma_i\}_{i=1}^{m'}$ to generate $m'$ new atoms, which are vector-valued $Q_{\max}$-bounded measurable functions from $\mathcal{X} \times \mathcal{A}$ to $\mathbb{R}^{|\mathcal{A}|}$, to be added to the learned dictionary $\mathcal{B}'_k$. The link functions extract "interesting" aspects of $Q_{k+1}$, potentially by considering the current dictionary $\mathcal{B}_k \bigcup \mathcal{B}'_k$. VPI is quite flexible in how the new atoms are generated and how large $m'$ can be. The theory allows $m'$ to be in the order of $n^a$ ($a > 0$), so one may add many potentially useful atoms without much deterioration in the performance. Regarding the choice of the link functions, the theory requires that at least $Q_{k+1}$ itself is being added to the dictionary, but it leaves other possibilities open. For example, one might apply nonlinearities (e.g., sigmoid functions) to $Q_{k+1}$. Or one might add atoms localized in parts of the state-action space with high residual errors – a heuristic which has been used previously in basis function construction. This procedure continues for $K$ iterations and the outcome will be $Q_K$. In the next section, we study the theoretical properties of the greedy policy w.r.t. $Q_K$, i.e., $\pi_K = \hat{\pi}(\cdot; Q_K)$.

*Remark* 1 (Comparison of VPI with FQI). Both VPI and FQI are indeed instances of AVI. If we compare VPI with the conventional implementation of FQI that uses a fixed set of linear basis

functions, we observe that FQI is the special case of VPI in which all atoms in the dictionary are used in the estimation. As VPI has a model selection step, its chosen estimator is not worse than FQI's (up to a small extra risk) and is possibly much better if the target is sparse in the dictionary. Moreover, extending the dictionary decreases the function approximation error with negligible effect on the model selection error. The same arguments apply to many other FQI versions that use a fixed data-independent set of basis functions and do not perform model selection.

## 4 Theoretical Analysis

In this section, we first study how the function approximation error propagates in VPI (Section 4.1) and then provide a finite-sample error upper bound as Theorem 3 in Section 4.2. All the proofs are in the supplementary material.

### 4.1 Propagation of Function Approximation Error

In this section, we present tools to upper bound the function approximation error at each iteration.

**Definition 2** (Concentrability Coefficient of Function Approximation Error Propagation). *(I) Let $\nu$ be a distribution over the state-action pairs, $(X, A) \sim \nu$, $\nu_{\mathcal{X}}$ be the marginal distribution of $X$, and $\pi_b(\cdot|\cdot)$ be the conditional probability of $A$ given $X$. Further, let $P$ be a transition probability kernel $P : \mathcal{X} \times \mathcal{A} \to \mathcal{M}(X)$ and $P_{x,a} = P(\cdot|x, a)$. Define the concentrability coefficient of one-step transitions w.r.t. $\nu$ by $C_{\nu \to \infty} = \left( \mathbb{E} \left[ \sup_{(y,a') \in \mathcal{X} \times \mathcal{A}} \left| \frac{1}{\pi_b(a'|y)} \frac{dP_{X,A}}{d\nu_{\mathcal{X}}}(y) \right| \right] \right)^{\frac{1}{2}}$, where $C_{\nu \to \infty} = \infty$ if $P_{x,a}$ is not absolutely continuous w.r.t. $\nu_{\mathcal{X}}$ for some $(x, a) \in \mathcal{X} \times \mathcal{A}$, or if $\pi_b(a'|y) = 0$ for some $(y, a') \in \mathcal{X} \times \mathcal{A}$. (II) Furthermore, for an optimal policy $\pi^*$ and an integer $m \geq 0$, let $\nu(P^{\pi^*})^m \in \mathcal{M}(\mathcal{X} \times \mathcal{A})$ denote the future state-action distribution obtained after $m$-steps of following $\pi^*$. Define $c_\nu(m) \triangleq \| \frac{d(\nu(P^{\pi^*})^m)}{d\nu} \|_\infty$. If the future state-action distribution $\nu(P^{\pi^*})^m$ is not absolutely continuous w.r.t. $\nu$, we let $c_\nu(m) = \infty$.*

The constant $C_{\nu \to \infty}$ is large if after transition step, the future states can be highly concentrated at some states where the probability of taking some action $a'$ is small or $d\nu_{\mathcal{X}}$ is small. Hence, the name "concentrability of one-step transitions". The definition of $C_{\nu \to \infty}$ is from Chapter 5 of Farahmand [16]. The constant $c_\nu(m)$ shows how much we deviate from $\nu$ whenever we follow an optimal policy $\pi^*$. It is notable that if $\nu$ happens to be the stationary distribution of the optimal policy $\pi^*$ (e.g., the samples are generated by an optimal expert), $c_\nu(m) = 1$ for all $m \geq 0$.

We now provide the following result that upper bounds the error caused by using $Q_k$ (which is the newly added atom to the dictionary) to approximate $T^*Q_k$. The proof is provided in the supplementary material.

**Lemma 1.** *Let $(Q_i)_{i=0}^k \subset B(\mathcal{X} \times \mathcal{A}, Q_{max})$ be a $Q_{max}$-bounded sequence of measurable action-value functions. Define $\varepsilon_i \triangleq T^*Q_i - Q_{i+1}$ ($0 \leq i \leq k-1$). Then, $\|Q_k - T^*Q_k\|_\nu^2 \leq \frac{(1+\gamma C_{\nu \to \infty})^2}{1-\gamma} \left[ \sum_{i=0}^{k-1} \gamma^{k-1-i} c_\nu(k-1-i) \|\varepsilon_i\|_\nu^2 + \gamma^k (2Q_{max})^2 \right]$.*

If there was no error at earlier iterations (i.e., $\|\varepsilon_i\|_\nu = 0$ for $0 \leq i \leq k-1$), the error $\|Q_k - T^*Q_k\|_\nu^2$ would be $O(\gamma^k Q_{max}^2)$, which is decaying toward zero with a geometrical rate. This is similar to the behaviour of the exact VI, i.e., $\|T^*Q_k - Q_k\|_\infty \leq (1+\gamma)\gamma^k \|Q^* - Q_0\|_\infty$.

The following result is Theorem 5.3 of Farahmand [16]. For the sake of completeness, we provide the proof in the supplementary material.

**Theorem 2.** *Let $(Q_k)_{k=0}^{k-1}$ be a sequence of state-action value functions and define $\varepsilon_i \triangleq T^*Q_i - Q_{i+1}$ ($0 \leq i \leq k$). Let $\mathcal{F}^{|\mathcal{A}|} : \mathcal{X} \times \mathcal{A} \to \mathbb{R}^{|\mathcal{A}|}$ be a subset of vector-valued measurable functions. Then, $\inf_{Q' \in \mathcal{F}^{|\mathcal{A}|}} \|Q' - T^*Q_k\|_\nu \leq \inf_{Q' \in \mathcal{F}^{|\mathcal{A}|}} \|Q' - (T^*)^{(k+1)}Q_0\|_\nu + \sum_{i=0}^{k-1} (\gamma C_{\nu \to \infty})^{k-i} \|\varepsilon_i\|_\nu$.*

This result quantifies the behaviour of the function approximation error and relates it to the function approximation error of approximating $(T^*)^{k+1}Q_0$ (which is a deterministic quantity depending only on the MDP itself, the function space $\mathcal{F}^{|\mathcal{A}|}$, and $Q_0$) and the errors of earlier iterations. This allows

us to provide a tighter upper bound for the function approximation error compared to the so-called *inherent Bellman error* $\sup_{Q\in\mathcal{F}^{|\mathcal{A}|}}\inf_{Q'\in\mathcal{F}^{|\mathcal{A}|}}\|Q'-T^*Q\|_\nu$ introduced by Munos and Szepesvári [17], whenever the errors at previous iterations are small.

## 4.2 Finite Sample Error Bound for VPI

In this section, we provide an upper bound on the performance loss $\|Q^*-Q^{\pi_K}\|_{1,\rho}$. This performance loss indicates the regret of following the policy $\pi_K$ instead of an optimal policy when the initial state-action is distributed according to $\rho$. We define the following concentrability coefficients similar to Farahmand et al. [18].

**Definition 3** (Expected Concentrability of the Future State-Action Distribution). *Given* $\rho,\nu \in \mathcal{M}(\mathcal{X}\times\mathcal{A})$, $m \geq 0$, *and an arbitrary sequence of stationary policies* $(\pi_m)_{m\geq 1}$, *let* $\rho P^{\pi_1}P^{\pi_2}\dots P^{\pi_m} \in \mathcal{M}(\mathcal{X}\times\mathcal{A})$ *denote the future state-action distribution obtained after* $m$ *transitions, when the first state-action pair is distributed according to* $\rho$ *and then we follow the sequence of policies* $(\pi_k)_{k=1}^m$. *For integers* $m_1,m_2 \geq 1$, *policy* $\pi$ *and the sequence of policies* $\pi_1,\dots,\pi_k$ *define the concentrability coefficients*

$$c_{VI_1,\rho,\nu}(m_1,m_2;\pi) \triangleq \left(\mathbb{E}\left[\left|\frac{d\left(\rho(P^\pi)^{m_1}(P^{\pi^*})^{m_2}\right)}{d\nu}(X,A)\right|^2\right]\right)^{\frac{1}{2}} \; and \; c_{VI_2,\rho,\nu}(m_1;\pi_1,\dots,\pi_k) \triangleq$$

$$\left(\mathbb{E}\left[\left|\frac{d(\rho(P^{\pi_k})^{m_1}P^{\pi_{k-1}}P^{\pi_{k-2}}\dots P^{\pi_1})}{d\nu}(X,A)\right|^2\right]\right)^{\frac{1}{2}}, \; where \; (X,A)\sim\nu. \; If \; the \; future \; state-action \; distribution \; \rho(P^\pi)^{m_1}(P^{\pi^*})^{m_2}$$ *(similarly, if* $\rho(P^{\pi_k})^{m_1}P^{\pi_{k-1}}P^{\pi_{k-2}}\cdots P^{\pi_1}$*) is not absolutely continuous w.r.t.* $\nu$, *we let* $c_{VI_1,\rho,\nu}(m_1,m_2;\pi)=\infty$ *(similarly,* $c_{VI_2,\rho,\nu}(m_1;\pi_1,\dots,\pi_k)=\infty$*).*

**Assumption A1** We make the following assumptions:
- For all values of $0 \leq k \leq K-1$, the dataset used by VPI at each iteration is $\mathcal{D}_n^{(k)} = \{(X_i^{(k)},A_i^{(k)},R_i^{(k)},X_i'^{(k)})\}_{i=1}^n$ with independent and identically distributed (i.i.d.) samples $(X_i^{(k)},A_i^{(k)}) \sim \nu \in \mathcal{M}(\mathcal{X}\times\mathcal{A})$ and $X_i'^{(k)} \sim P(\cdot|X_i^{(k)},A_i^{(k)})$ and $R_i^{(k)} \sim \mathcal{R}(\cdot,\cdot|X_i^{(k)},A_i^{(k)})$ for $i=1,2,\dots,n$.
- For $1 \leq k,k' \leq K-1$ and $k \neq k'$, the datasets $\mathcal{D}_n^{(k)}$ and $\mathcal{D}_n^{(k')}$ are independent.
- There exists a constant $Q_{\max} \geq 1$ such that for any $Q \in B(\mathcal{X}\times\mathcal{A};Q_{\max})$, $|\hat{T}^*Q(X,A)| \leq Q_{\max}$ almost surely (a.s.).
- For all $g \in \mathcal{B}_0$, $\|g\|_\infty \leq L < \infty$.
- The number of atoms $m$ used from the dictionary $\mathcal{B}_0$ is $m = \lceil n^a \rceil$ for some finite $a > 0$. The number of link functions $m'$ used at each iteration is at most $m/K$.
- At iteration $k$, each of the link functions $\{\sigma_i\}_{i=1}^{m'}$ maps $\beta_{Q_{\max}}Q_{k+1}$ and the dictionary $\mathcal{B}_k \bigcup \mathcal{B}_k'$ to an element of the space of vector-valued $Q_{\max}$-bounded measurable functions $\mathcal{X}\times\mathcal{A}\to\mathbb{R}^{|\mathcal{A}|}$. At least one of the mappings returns $\beta_{Q_{\max}}Q_{k+1}$.

Most of these assumptions are mild and some of them can be relaxed. The i.i.d. assumption can be relaxed using the so called *independent block technique* [19], but it results in much more complicated proofs. We conjecture that the independence of datasets at different iterations might be relaxed as well under certain condition on the Bellman operator (cf. Section 4.2 of [17]). The condition on the number of atoms $m$ and the number of link functions being polynomial in $n$ are indeed very mild.

In order to compactly present our result, we define $a_k = \frac{(1-\gamma)\gamma^{K-k-1}}{1-\gamma^{K+1}}$ for $0 \leq k < K$. Note that the behaviour of $a_k \propto \gamma^{K-k-1}$, so it gives more weight to later iterations. Also define $C_1(k) \triangleq \sum_{i=0}^{k-1}\gamma^{k-i}C_{\nu\to\infty}^{2(k-i)}$ $(k=1,2,\dots)$ and $C_2 \triangleq \frac{(1+\gamma C_{\nu\to\infty})^2}{1-\gamma}$. For $0 \leq s \leq 1$, define

$$C_{VI,\rho,\nu}(K;s) =$$

$$(\frac{1-\gamma}{2})^2 \sup_{\pi_1',\dots,\pi_K'} \sum_{k=0}^{K-1} a_k^{2(1-s)}\left[\sum_{m\geq 0}\gamma^m\left(c_{VI_1,\rho,\nu}(m,K-k;\pi_K') + c_{VI_2,\rho,\nu}(m+1;\pi_{k+1}',\dots,\pi_K')\right)\right]^2,$$

where in the last definition the supremum is taken over all policies. The following theorem is the main theoretical result of this paper. Its proof is provided in the supplementary material.

**Theorem 3.** *Consider the sequence $(Q_k)_{k=0}^K$ generated by VPI (Algorithm 1). Let Assumptions A1 hold. For any fixed $0 < \delta < 1$, recursively define the sequence $(b_i)_{i=0}^K$ as follows:*

$$b_0^2 \triangleq c_1 Q_{max}^3 \sqrt{\frac{\log\left(\frac{nK}{\delta}\right)}{n}} + 3 \inf_{Q' \in B_{Q_{max}}(\mathcal{L}_1(\mathcal{B}_{0,m};\nu))} \left\|Q' - T^* Q_0\right\|_\nu^2,$$

$$b_k^2 \triangleq c_2 Q_{max}^3 \sqrt{\frac{\log\left(\frac{nK}{\delta}\right)}{n}} +$$

$$c_3 \min \left\{ \inf_{Q' \in B_{Q_{max}}(\mathcal{L}_1(\mathcal{B}_{0,m};\nu))} \left\|Q' - (T^*)^{k+1} Q_0\right\|_\nu^2 + C_1(k) \sum_{i=0}^{k-1} \gamma^{k-i} b_i^2, \right.$$

$$\left. C_2 \left( \sum_{i=0}^{k-1} \gamma^{k-1-i} c_\nu(k-1-i) \, b_i^2 + \gamma^k (2Q_{max})^2 \right) \right\}, \quad (k \geq 1)$$

*for some $c_1, c_2, c_3 > 0$ that are only functions of $Q_{max}$ and $L$. Then, for any $k = 0, 1, \dots, K-1$, it holds that $\left\|Q_{k+1} - T^* Q_k\right\|_\nu^2 \leq b_k^2$, with probability at least $1 - \frac{k\delta}{K}$. Furthermore, define the discounted sum of errors as $\mathcal{E}(s) \triangleq \sum_{k=0}^{K-1} a_k^{2s} b_k$ (for $s \in [0,1]$). Choose $\rho \in \mathcal{M}(\mathcal{X} \times \mathcal{A})$. The $\rho$-weighted performance loss of $\pi_K$ is upper bounded as*

$$\left\|Q^* - Q^{\pi_K}\right\|_{1,\rho} \leq \frac{2\gamma}{(1-\gamma)^2} \left[ \inf_{s \in [0,1]} C_{VI,\rho,\nu}^{1/2}(K;s) \mathcal{E}^{1/2}(s) + 2\gamma^K Q_{max} \right],$$

*with probability at least $1 - \delta$.*

The value of $b_k$ is a deterministic upper bound on the error $\left\|Q_{k+1} - T^* Q_k\right\|_\nu$ of each iteration of VPI. We would like $b_k$ to be close to zero, because the second part of the theorem implies that $\left\|Q^* - Q^{\pi_K}\right\|_{1,\rho}$ would be small too. If we study $b_k^2$, we observe two main terms: The first term, which behaves as $\sqrt{\frac{\log(nK/\delta)}{n}}$, is the estimation error. The second term describes the function approximation error. For $k \geq 1$, it consists of two terms from which the minimum is selected. The first term inside $\min\{\cdot, \cdot\}$ describes the behaviour of the function approximation error when we only use the predefined dictionary $\mathcal{B}_{0,m}$ to approximate $T^* Q_k$ (see Theorem 2). The second term describes the behaviour of the function approximation error when we only consider $Q_k$ as the approximant of $T^* Q_k$ (see Lemma 1). The error caused by this approximation depends on the error made in earlier iterations. The current analysis only considers the atom $Q_k$ from the learned dictionary, but VPI may actually use other atoms to represent $T^* Q_k$. This might lead to much smaller function approximation errors. Hence, our analysis shows that in terms of function approximation error, our method is sound and superior to not increasing the size of the dictionary. However, revealing the full power of VPI remains as future work. Just as an example, if $\mathcal{B}_0$ is complete in $L_2(\nu)$, by letting $n, m \to \infty$ both the estimation error and function approximation error goes to zero and the method is consistent and converges to the optimal value function.

## 5   Conclusion

This work introduced VPI, an approximate value iteration algorithm that aims to find a close to optimal policy using a dictionary of atoms (or features). The VPI algorithm uses a modified Orthogonal Matching Pursuit that is equipped with a model selection procedure. This allows VPI to find a sparse representation of the value function in large, and potentially overcomplete, dictionaries. We theoretically analyzed VPI and provided a finite-sample error upper bound for it. The error bound shows the effect of the number of samples as well as the function approximation properties of the predefined dictionary, and the effect of learned atoms.

This paper is a step forward to better understanding how overcomplete dictionaries and sparsity can effectively be used in the RL/Planning context. A more complete theory describing the effect of adding atoms to the dictionary remains to be established. We are also planning to study VPI's empirical performance, and comparing with other feature construction methods. We note that our main focus was on the statistical properties of the algorithm, not on computational efficiency; optimizing computation speed will be an interesting topic for future investigation.

## Footnotes

[1] From the statistical viewpoint and ignoring the computational difficulty of working with large dictionaries.

[2] The notation will be defined precisely in Section 2.

[3]The value of $c_1$ depends only on $Q_{\max}$ and $a$. We do not explicitly specify it since the value that is determined by the theory shall be quite conservative. One may instead find it by the trial and error. Moreover, in practice we may stop much earlier than $n$ iterations.

[4]When the number of atoms is larger than the number of samples ($i > n$), one may use the Moore–Penrose pseudoinverse to perform the orthogonal projection.

# References

[1] Sridhar Mahadevan and Mauro Maggioni. Proto-value functions: A Laplacian framework for learning representation and control in markov decision processes. *Journal of Machine Learning Research*, 8: 2169–2231, 2007. 1

[2] Ronald Parr, Christopher Painter-Wakefield, Lihong Li, and Michael Littman. Analyzing feature generation for value-function approximation. In *ICML '07: Proceedings of the 24th international conference on Machine learning*, pages 737 – 744, New York, NY, USA, 2007. ACM. 1

[3] Amir-massoud Farahmand, Mohammad Ghavamzadeh, Csaba Szepesvári, and Shie Mannor. Regularized policy iteration. In D. Koller, D. Schuurmans, Y. Bengio, and L. Bottou, editors, *Advances in Neural Information Processing Systems (NIPS - 21)*, pages 441–448. MIT Press, 2009. 1

[4] Amir-massoud Farahmand, Mohammad Ghavamzadeh, Csaba Szepesvári, and Shie Mannor. Regularized fitted Q-iteration for planning in continuous-space Markovian Decision Problems. In *Proceedings of American Control Conference (ACC)*, pages 725–730, June 2009. 1, 4

[5] Gavin Taylor and Ronald Parr. Kernelized value function approximation for reinforcement learning. In *ICML '09: Proceedings of the 26th Annual International Conference on Machine Learning*, pages 1017–1024, New York, NY, USA, 2009. ACM. 1

[6] J. Zico Kolter and Andrew Y. Ng. Regularization and feature selection in least-squares temporal difference learning. In *ICML '09: Proceedings of the 26th Annual International Conference on Machine Learning*, pages 521–528. ACM, 2009. 1

[7] Jeff Johns, Christopher Painter-Wakefield, and Ronald Parr. Linear complementarity for regularized policy evaluation and improvement. In J. Lafferty, C. K. I. Williams, J. Shawe-Taylor, R.S. Zemel, and A. Culotta, editors, *Advances in Neural Information Processing Systems (NIPS - 23)*, pages 1009–1017. 2010. 1

[8] Mohammad Ghavamzadeh, Alessandro Lazaric, Rémi Munos, and Matthew Hoffman. Finite-sample analysis of lasso-TD. In Lise Getoor and Tobias Scheffer, editors, *Proceedings of the 28th International Conference on Machine Learning (ICML-11)*, ICML '11, pages 1177–1184, New York, NY, USA, June 2011. ACM. ISBN 978-1-4503-0619-5. 1

[9] Y. C. Pati, R. Rezaiifar, and P. S. Krishnaprasad. Orthogonal matching pursuit: Recursive function approximation with applications to wavelet decomposition. In *Proceedings of the 27th Annual Asilomar Conference on Signals, Systems, and Computers*, pages 40–44, 1993. 1

[10] Geoffrey M. Davis, Stéphane Mallat, and Marco Avellaneda. Adaptive greedy approximations. *Journal of Constructive Approximation*, 13:57–98, 1997. 1

[11] Jeff Johns. *Basis Construction and Utilization for Markov Decision Processes using Graphs*. PhD thesis, University of Massachusetts Amherst, 2010. 1

[12] Christopher Painter-Wakefield and Ronald Parr. Greedy algorithms for sparse reinforcement learning. In *Proceedings of the 29th International Conference on Machine Learning (ICML) (Accepted)*, 2012. 1

[13] Damien Ernst, Pierre Geurts, and Louis Wehenkel. Tree-based batch mode reinforcement learning. *Journal of Machine Learning Research*, 6:503–556, 2005. 2, 4

[14] Csaba Szepesvári. *Algorithms for Reinforcement Learning*. Morgan Claypool Publishers, 2010. 2

[15] Andrew R. Barron, Albert Cohen, Wolfgang Dahmen, and Ronald A. Devore. Approximation and learning by greedy algorithms. *The Annals of Statistics*, 36(1):64–94, 2008. 3, 4

[16] Amir-massoud Farahmand. *Regularization in Reinforcement Learning*. PhD thesis, University of Alberta, 2011. 6

[17] Rémi Munos and Csaba Szepesvári. Finite-time bounds for fitted value iteration. *Journal of Machine Learning Research*, 9:815–857, 2008. 7

[18] Amir-massoud Farahmand, Rémi Munos, and Csaba Szepesvári. Error propagation for approximate policy and value iteration. In J. Lafferty, C. K. I. Williams, J. Shawe-Taylor, R.S. Zemel, and A. Culotta, editors, *Advances in Neural Information Processing Systems (NIPS - 23)*, pages 568–576. 2010. 7

[19] Bin Yu. Rates of convergence for empirical processes of stationary mixing sequences. *The Annals of Probability*, 22(1):94–116, January 1994. 7

